# Time-Varying Dynamic Bayesian Networks

**Le Song, Mladen Kolar and Eric P. Xing**
School of Computer Science, Carnegie Mellon University
{lesong, mkolar, epxing}@cs.cmu.edu

## Abstract

Directed graphical models such as Bayesian networks are a favored formalism for modeling the dependency structures in complex multivariate systems such as those encountered in biology and neural science. When a system is undergoing dynamic transformation, temporally rewiring networks are needed for capturing the dynamic causal influences between covariates. In this paper, we propose time-varying dynamic Bayesian networks (TV-DBN) for modeling the structurally varying directed dependency structures underlying non-stationary biological/neural time series. This is a challenging problem due the non-stationarity and sample scarcity of time series data. We present a kernel reweighted $\ell_1$-regularized auto-regressive procedure for this problem which enjoys nice properties such as computational efficiency and provable asymptotic consistency. To our knowledge, this is the first practical and statistically sound method for structure learning of TV-DBNs. We applied TV-DBNs to time series measurements during yeast cell cycle and brain response to visual stimuli. In both cases, TV-DBNs reveal interesting dynamics underlying the respective biological systems.

## 1   Introduction

Analysis of biological networks has led to numerous advances in understanding the organizational principles and functional properties of various biological systems, such as gene regulatory systems [1] and central nervous systems [2]. However, most such results are based on *static networks*, that is, networks with invariant topology over a given set of biological entities. A major challenge in systems biology is to understand and model, quantitatively, the dynamic topological and functional properties of biological networks. We refer to these time or condition specific biological circuitries as *time-varying networks or structural non-stationary networks*, which are ubiquitous in biological systems. For example (*i*) over the course of a cell cycle, there may exist multiple biological "themes" that determine functions of each gene and their regulatory relations, and these "themes" are dynamic and stochastic. As a result, the molecular networks at each time point are context-dependent and can undergo systematic rewiring rather than being invariant over time [3]. (*ii*) The emergence of a unified cognitive moment relies on the coordination of scattered mosaics of functionally specialized brain regions. Neural assemblies, distributed local networks of neurons transiently linked by dynamic connections, enable the emergence of coherent behaviour and cognition [4].

A key technical hurdle preventing us from an in-depth investigation of the mechanisms that drive temporal biological processes is the unavailability of *serial snapshots* of time-varying networks underlying biological processes. Current technology does not allow for experimentally determining a series of time specific networks for a realistic dynamic biological system. Usually, only time series measurements of the activities of the nodes can be made, such as microarray, EEG or fMRI. Our goal is to recover the latent time-varying networks underlying biological processes, with temporal resolution up to every single time point based on time series measurements of the nodal states. Recently, there has been a surge of interests along this direction [5, 6, 7, 8, 9, 10]. However, most existing approaches are computationally expensive, making large scale genome-wide reverse engineering nearly infeasible. Furthermore, these methods also lack formal statistical characterization of

the estimation procedure. For instance, non-stationary dynamic Bayesian networks are introduced in [9], where the structures are learned via MCMC sampling; such approach is not likely to scale up to more than 1000 nodes and without a regularization term it is also prone to overfitting when the dimension of the data is high but the number of observations is small. More recent efforts have focused on efficient kernel reweighted or total-variation penalized sparse structure recovery methods for *undirected* time-varying networks [10, 11, 12], which possess both attractive computational schemes and rigorous statistical consistency results. However, what has not been addressed so far is how to recover *directed* time-varying networks. Our current paper advances in this direction.

More specifically, we propose time-varying dynamic Bayesian networks (TV-DBN) for modeling the directed time-evolving network structures underlying non-stationary biological time series. To make this problem statistically tractable, we rely on the assumption that the underlying network structures are sparse and vary smoothly across time. We propose a kernel reweighted $\ell_1$-regularized auto-regressive approach for learning this sequence of networks. Our approach has the following attractive properties: (*i*) The aggregation of observations from adjacent time points by kernel reweighting greatly alleviates the statistical problem of sample scarcity when the networks can change at each time point whereas only one or a few time series replicates are available. (*ii*) The problem of structural estimation for a TV-DBN decomposes into a collection of simpler and atomic structural learning problems. We can choose from a battery of highly scalable $\ell_1$-regularized least-square solvers for learning each structure. (*iii*) We can formally characterize the conditions under which our estimation procedure is structurally consistent: as time series are sampled in increasing resolution, our algorithm can recover the true structure of the underlying TV-DBN with high probability.

It is worth emphasizing that our approach is very different from earlier approaches, such as the structure learning algorithms for dynamic Bayesian networks [13], which learn time-homogeneous dynamic systems with fixed node dependencies, or approaches which start from an *a priori* static network and then trace time-dependent activities [3]. The Achilles' heel of this latter approach is that edges that are transient over a short period of time may be missed by the summary static network in the first place. Furthermore, our approach is also different from change point based algorithms [14, 8] which first segment time series and then fit an invariant structure to each segment. These approaches can only recover piece-wise stationary models rather than constantly varying networks. In our experiments, we demonstrate the advantange of TV-DBNs using synthetic networks. We also apply TV-DBNs to real world datasets: a gene expression dataset measured during yeast cell cycle; and an EEG dataset recorded during a motor imagination task. In both cases, TV-DBNs reveal interesting time-varying causal structures of the underlying biological systems.

## 2   Preliminary

We concern ourselves with stochastic processes in time or space domains, such as the dynamic control of gene expression during cell cycle, or the sequential activation of brain areas during cognitive decision making, of which the state of a variable at one time point is determined by the states of a set of variables at previous time points. Models describing a stochastic *temporal* processes can be naturally represented as *dynamic Bayesian networks* (DBN) [15]. Taking the transcriptional regulation of gene expression as an example, let $\boldsymbol{X}^t := (X_1^t, \ldots, X_p^t)^\top \in \mathbb{R}^p$ be a vector representing the expression levels of $p$ genes at time $t$, a stochastic dynamic process can be modeled by a "first-order Markovian transition model" $p(\boldsymbol{X}^t|\boldsymbol{X}^{t-1})$, which defines the probabilistic distribution of gene expressions at time $t$ given those at time $t-1$. Under this assumption, the likelihood of the observed expression levels of these genes over a time series of $T$ steps can be expressed as:

$$p(\boldsymbol{X}^1, \ldots, \boldsymbol{X}^T) = p(\boldsymbol{X}^1) \prod_{t=2}^{T} p(\boldsymbol{X}^t|\boldsymbol{X}^{t-1}) = p(\boldsymbol{X}^1) \prod_{t=2}^{T} \prod_{i=1}^{p} p(X_i^t|\boldsymbol{X}_{\pi_i}^{t-1}), \quad (1)$$

where we assume that the topology of the networks is specified by a set of regulatory relations $\boldsymbol{X}_{\pi_i}^{t-1} := \{X_j^{t-1} : X_j^{t-1} \text{ regulates } X_i^t\}$, and hence the transition model $p(\boldsymbol{X}^t|\boldsymbol{X}^{t-1})$ factors over individual genes, *i.e.*, $\prod_i p(X_i^t|\boldsymbol{X}_{\pi_i}^{t-1})$. Each $p(X_i^t|\boldsymbol{X}_{\pi_i}^{t-1})$ can be viewed as a regulatory gate function that takes multiple covariates (regulators) and produce a single response.

A simple form of the transition model $p(\boldsymbol{X}^t|\boldsymbol{X}^{t-1})$ in a DBN is a *linear dynamics model*:

$$\boldsymbol{X}^t = \boldsymbol{A} \cdot \boldsymbol{X}^{t-1} + \boldsymbol{\epsilon}, \quad \boldsymbol{\epsilon} \sim \mathcal{N}(\boldsymbol{0}, \sigma^2 \boldsymbol{I}), \quad (2)$$

where $\boldsymbol{A} \in \mathbb{R}^{p \times p}$ is a matrix of coefficients relating the expressions at time $t-1$ to those of the next time point, and $\boldsymbol{\epsilon}$ is a vector of isotropic zero mean Gaussian noise with variance $\sigma^2$. In this

case, the gate function that defines the conditional distribution $p(X_i^t | \boldsymbol{X}_{\pi_i}^{t-1})$ can be expressed as a univariate Gaussian, *i.e.*, $p(X_i^t | \boldsymbol{X}_{\pi_i}^{t-1}) = \mathcal{N}(X_i^t; \boldsymbol{A}_{i\cdot} \boldsymbol{X}^{t-1}, \sigma^2)$, where $\boldsymbol{A}_{i\cdot}$ denotes the $i^{\text{th}}$ row of the matrix $\boldsymbol{A}$. This model is also known as an *auto-regressive* model.

The major reason for favoring DBNs over standard Bayesian networks (BN) or undirected graphical models is its enhanced semantic interpretability. An edge in a BN does not necessarily imply causality due to the *Markov equivalence* of different edge configurations in the network [16]. In DBNs (of the type defined above), all directed edges only point from time $t-1$ to $t$, which bear a natural causal implication and are more likely to suggest regulatory relations. The auto-regressive model in (2) also offers an elegant formal framework for consistent estimation of the structures of DBNs; we can read off the edges between variables in $\boldsymbol{X}^{t-1}$ and $\boldsymbol{X}^t$ by simply identifying the nonzero entries in the transition matrix $\boldsymbol{A}$. For example, the non-zero entries of $\boldsymbol{A}_{i\cdot}$ represent the set of regulator $\boldsymbol{X}_{\pi_i}$ that directly lead to a response on $X_i$.

Contrary to the name of dynamic Bayesian networks may suggest, DBNs are *time-invariant* models and the underlying network structures do *not* change over time. That is, the dependencies between variables in $\boldsymbol{X}^{t-1}$ and $\boldsymbol{X}^t$ are fixed, and both $p(\boldsymbol{X}^t | \boldsymbol{X}^{t-1})$ and $\boldsymbol{A}$ are invariant over time. The term "dynamic" only means that the DBN can model dynamical systems. In the sequel, we will present a new formalism where the structures of DBNs are time-varying rather than invariant.

## 3  A New Formalism: Time-Varying Dynamic Bayesian Networks

We will focus on recovering the *directed* time-varying network structure (or the locations of non-zero entries in $\boldsymbol{A}$) rather than the exact edge values. This is related to the structure estimation problems studied in [11, 12], but in our case for auto-regressive models (and hence *directed* networks). Structure estimation results in parse models for easy interpretation, but it is statistically more challenging than the value estimation problem. This is also different from estimating a non-stationary model in the conventional sense, where one interests in recovering the exact values of the varying coefficients [17, 18]. To make this distinction clear, we use the following 3 examples:

$$B_1 = \begin{pmatrix} 0 & 1 & 0 \\ 0 & 0 & 1 \\ 0 & 0 & 0 \end{pmatrix}, \qquad B_2 = \begin{pmatrix} 0 & 0.1 & 0 \\ 0 & 0 & 3 \\ 0 & 0 & 0 \end{pmatrix}, \qquad B_3 = \begin{pmatrix} 0 & 1 & 0.1 \\ 0 & 0 & 1.1 \\ 0 & 0.1 & 0 \end{pmatrix}. \tag{3}$$

Matrices $B_1$ and $B_2$ encode the same graph structure, since the locations of their non-zero entries are exactly same. Although $B_1$ is closer to $B_3$ than $B_2$ in terms of matrix values (eg. measured in Frobenius norm), they encodes very different graph strucutres.

Formally, let graph $\mathcal{G}^t = (\mathcal{V}, \mathcal{E}^t)$ represents the conditional independence relations between the components of random vectors $\boldsymbol{X}^{t-1}$ and $\boldsymbol{X}^t$. The vertex set $\mathcal{V}$ is a common set of variables underlying $\boldsymbol{X}^{1:T}$, *i.e.*, each node in $\mathcal{V}$ corresponds to a sequence of variables $X_i^{1:T}$. The edge set $\mathcal{E}^t \subseteq \mathcal{V} \times \mathcal{V}$ contains directed edges from components of $\boldsymbol{X}^{t-1}$ to those of $\boldsymbol{X}^t$; an edge $(i,j) \notin \mathcal{E}^t$ if and only if $X_i^t$ is conditionally independent of $X_j^{t-1}$ given the rest of the variables in the model. Due to the time-varying nature of the networks, the transition model $p^t(\boldsymbol{X}^t | \boldsymbol{X}^{t-1})$ in (1) becomes time dependent. In the case of the auto-regressive DBN in (2), its time-varying extension becomes:

$$\boldsymbol{X}^t = \boldsymbol{A}^t \cdot \boldsymbol{X}^{t-1} + \boldsymbol{\epsilon}, \quad \boldsymbol{\epsilon} \sim \mathcal{N}(\boldsymbol{0}, \sigma^2 \boldsymbol{I}), \tag{4}$$

and our goal is to estimate the non-zero entries in the sequence of time dependent transition matrices $\{\boldsymbol{A}^t\}$ ($t = 1 \ldots T$). The directed edges $\mathcal{E}^t := \mathcal{E}^t(\boldsymbol{A}^t)$ in network $\mathcal{G}^t$ associated with each $\boldsymbol{A}^t$ can be recovered via $\mathcal{E}^t = \{(i,j) \in \mathcal{V} \times \mathcal{V} \mid i \neq j, \boldsymbol{A}_{ij}^t \neq 0\}$.

## 4  Estimating Time-Varying DBN

Note that if we follow the naive assumption that each temporal snapshot is a completely different network, the task of jointly estimating $\{\boldsymbol{A}^t\}$ by maximizing the log-likelihood would be statistically impossible because the estimator would suffer from extremely high variance due to sample scarcity. Therefore, we make a statistically tractable yet realistic assumption that the underlying network structures are sparse and vary smoothly across time; and hence temporally adjacent networks are likely to share common edges than temporally distal networks.

Overall, we have designed a procedure that decomposes the problem of estimating the time-varying networks along two orthogonal axes. The first axis is along the time, where we estimate the network for each time point separately by reweighting the observations accordingly; and the second axis is along the set of genes, where we estimate the neighborhood for each gene separately and then join these neighborhoods to form the overall network. One benefit of such decomposition is that the estimation problem is reduced to a set of atomic optimizations, one for each node $i$ ($i = 1 \ldots |\mathcal{V}|$) at each time point $t^*$ ($t^* = 1 \ldots T$):

$$\hat{\boldsymbol{A}}_{i\cdot}^{t^*} = \operatorname*{argmin}_{\boldsymbol{A}_{i\cdot}^{t^*} \in \mathbb{R}^{1 \times n}} \frac{1}{T} \sum_{t=1}^{T} w^{t^*}(t)(x_i^t - \boldsymbol{A}_{i\cdot}^{t^*} \boldsymbol{x}^{t-1})^2 + \lambda \left\| \boldsymbol{A}_{i\cdot}^{t^*} \right\|_1, \qquad (5)$$

where $\lambda$ is a parameter for the $\ell_1$-regularization term, which controls the number of non-zero entries in the estimated $\hat{\boldsymbol{A}}_{i\cdot}^{t^*}$, and hence the sparsity of the networks; $w^{t^*}(t)$ is the weighting of an observation from time $t$ when we estimate the network at time $t^*$. More specifically, it is defined as $w^{t^*}(t) = \frac{K_h(t-t^*)}{\sum_{t=1}^{T} K_h(t-t^*)}$, where $K_h(\cdot) = K(\frac{\cdot}{h})$ is a symmetric nonnegative kernel function and $h$ is the kernel bandwidth. We use a Gaussian RBF kernel, $K_h(t) = \exp(-\frac{t^2}{h})$, in our later experiments. Note that multiple measurements at the same time point are considered as *i.i.d.* observations and can be trivially handled by assigning them the same weights.

The objective defined in (5) is essentially a weighted regression problem. The square loss function is due to the fact that we are fitting a linear model with uncorrelated Gaussian noise. Two other key components of the objective are: (*i*) a kernel reweighting scheme for aggregating observations across time; and (*ii*) an $\ell_1$-regularization for sparse structure estimation. The first component originates from our assumption that the structural changes of the network vary smoothly across time. This assumption allows us to borrow information across time by reweighting the observations from different time points and then treating them as if they were *i.i.d.* observations. Intuitively, the weighting should place more emphasis on observations at or near time point $t^*$ with weights becoming smaller as observations move further away from time point $t^*$. The second component is to promote sparse structure and avoid model overfitting. This is also consistent with the biological observation that networks underlying biological processes are parsimonious in structure. For example, a transcription factor only controls a small fraction of target genes at particular time point or under a specific condition [19]. It is well-known that $\ell_1$-regularized least square linear regression, has a parsimonious property, and exhibits model-selection consistency (i.e., recovers the set of true non-zero regression coefficients asymptotically) in noisy settings even when $p \gg T$ [20].

Note that our procedure can also be easily extended to learn the structure of auto-regressive models of higher order $D$: $\boldsymbol{X}^t = \sum_{d=1}^{D} \boldsymbol{A}^t(d) \cdot \boldsymbol{X}^{t-d} + \boldsymbol{\epsilon}, \quad \boldsymbol{\epsilon} \sim \mathcal{N}(\boldsymbol{0}, \sigma^2 \boldsymbol{I})$. The change we need to make is to incorporate the higher order auto-regressive coefficients in the square loss function, *i.e.*, $(x_i^t - \sum_{d=1}^{D} \boldsymbol{A}_{i\cdot}^{t^*}(d)\boldsymbol{x}^{t-d})^2$, and penalize the $\ell_1$-norms of these $\boldsymbol{A}_{i\cdot}^{t^*}(d)$ correspondingly.

# 5 Optimization

Estimating time-varying networks using the decomposition scheme above requires solving a collection of optimization problems in (5). In a genome-wide reverse engineering task, there can be tens of thousands of genes and hundreds of time points, so one can easily have a million optimization problems. Therefore, it is essential to use an efficient algorithm for solving the atomic optimization problem in (5), which can be trivially parallelized for each genes at each time point.

Instead of solving the form of the optimization problem in (5), we will push the weighting $w^{t^*}(t)$ into the square loss function by scaling the covariates and response variables by $\sqrt{w^{t^*}(t)}$, *i.e.* $\tilde{x}_i^t \leftarrow \sqrt{w^{t^*}(t)}x_i^t$ and $\tilde{\boldsymbol{x}}^{t-1} \leftarrow \sqrt{w^{t^*}(t)}\boldsymbol{x}^{t-1}$. After this transformation, the optimization problem becomes a standard $\ell_1$-regularized least-square problem which can be solved via a battery of highly scalable and specialized solvers, such as the shooting algorithm [21]. The shooting algorithm is a simple, straightforward and fast algorithm that iteratively solves a system of nonlinear equations related to the optimality condition of problem (5): $\frac{2}{T} \sum_{t=1}^{T} (\boldsymbol{A}_{i\cdot}^{t^*} \tilde{\boldsymbol{x}}^{t-1} - \tilde{x}_i^t)\tilde{x}_j^{t-1} = -\lambda \operatorname{sign}(\boldsymbol{A}_{ij}^{t^*})$ ($\forall j = 1 \ldots p$). At each iteration of the shooting algorithm, one entries of $\boldsymbol{A}_{i\cdot}$ is updated by holding all other entries fixed. Overall, our procedure for estimating time-varying networks is summarized in Algorithm 1, which uses the shooting algorithm as the key building block (step

---

**Algorithm 1**: Procedure for Estimating Time-Varying DBN

---

**Input**: Time series $\{\boldsymbol{x}^1, \ldots, \boldsymbol{x}^T\}$, regularization parameter $\lambda$ and kernel parameter $h$.
**Output**: Time-varying networks $\{\boldsymbol{A}^1, \ldots, \boldsymbol{A}^T\}$.

1 **begin**
2     Introduce variable $\boldsymbol{A}^0$ and randomly initialize it
3     **for** $i = 1 \ldots p$ **do**
4        **for** $t^* = 1 \ldots T$ **do**
5           Initialize: $\boldsymbol{A}_{i\cdot}^{t^*} \leftarrow \boldsymbol{A}_{i\cdot}^{t^*-1}$
6           Scale time series: $\tilde{x}_i^t \leftarrow \sqrt{w^{t^*}(t)}x_i^t, \ \tilde{\boldsymbol{x}}^{t-1} \leftarrow \sqrt{w^{t^*}(t)}\boldsymbol{x}^{t-1} \ (t = 1 \ldots T)$
7           **while** $\boldsymbol{A}_{i\cdot}^{t^*}$ *not converges* **do**
8              **for** $j = 1 \ldots p$ **do**
9                 Compute: $S_j \leftarrow \frac{2}{T}\sum_{t=1}^{T}(\sum_{k \neq j}\boldsymbol{A}_{ik}^{t^*}\tilde{x}_k^{t-1} - \tilde{x}_i^t)\tilde{x}_j^{t-1}, \ b_j = \frac{2}{T}\sum_{t=1}^{T}\tilde{x}_j^{t-1}\tilde{x}_j^{t-1}$
10                 Update: $\boldsymbol{A}_{ij}^{t^*} \leftarrow (\text{sign}(S_j - \lambda)\lambda - S_j)/b_j, \ \text{if } |S_j| > \lambda, \ \text{otherwise } 0$

11 **end**

---

7-10). In step 5, we uses a warm start for each atomic optimization problem: since the networks vary smoothly across time, we can use $\boldsymbol{A}_{i\cdot}^{t^*-1}$ as a good initialization for $\boldsymbol{A}_{i\cdot}^{t^*}$ for further speedup.

## 6   Statistical Properties

In this section, we study the statistical consistency of the estimation procedure in Section 4. Our analysis is different from the consistency results presented by [11] on recovering time-varying undirected graphical models. Their analysis deals with Frobenius norm consistency which is a weaker result than the structural consistency we pursue here. Our structural consistency result for TV-DBNs estimation procedure follows the proof strategy of [20]; however, the analysis is complicated by two major factors. First, times series observations are very often non-*i.i.d.*— current observations may depend on past history. Second, we are modeling non-stationary processes, where we need to deal with the additional bias term that arises due to locally stationary approximation to non-stationarity. In the following, we state our assumptions and theorem, but leave the detailed proof of this theorem for a full version of the paper (a sketch of the proof can be found in the appendix).

**Theorem 1** *Assume that the conditions below hold:*

1. *Elements of $\boldsymbol{A}^t$ are smooth functions with bounded second derivatives, i.e. there exists a constant $L > 0$ s.t. $|\frac{\partial}{\partial t}\boldsymbol{A}_{ij}^t| < L$ and $|\frac{\partial^2}{\partial t^2}\boldsymbol{A}_{ij}^t| < L$.*
2. *The minimum absolute value of non-zero elements of $\boldsymbol{A}^t$ is bounded away from zero at observation points, and this bound tends to zero as we observe more and more samples, i.e., $a_{\min} := \min_{t \in \{1/T, 2/T, \ldots, 1\}} \min_{i \in [p], j \in S_i^t} |A_{ij}^t| > 0$.*
3. *Let $\boldsymbol{\Sigma}^t = \mathbb{E}[\boldsymbol{X}^t(\boldsymbol{X}^t)^T] = [\sigma_{ij}(t)]_{i,j=1}^{p}$ and let $S_i^t$ denote the set of non-zero elements of the $i$-th row of the matrix $\boldsymbol{A}^t$, i.e. $S_i^t = \{j \in [p] \ : \ \boldsymbol{A}_{ij}^t \neq 0\}$. Assume that there exist a constant $d \in (0, 1]$ s.t. $\max_{j \in S_i^t, k \neq j} |\sigma_{jk}(t)| \leq \frac{d}{s}, \forall i \in [p], t \in [0, 1]$, where $s$ is an upper bound on the number of non-zero elements, i.e. $s = \max_{t \in [0,1]} \max_{i \in [p]} |S_i^t|$.*
4. *The kernel $K(\cdot) : \mathbb{R} \mapsto \mathbb{R}$ is a symmetric function and has bounded support on $[0, 1]$. There exists a constant $M_K$ s.t. $\max_{x \in \mathbb{R}} |K(x)| \leq M_K$ and $\max_{x \in \mathbb{R}} K(x)^2 \leq M_K$.*

*Let the regularization parameter scale as $\lambda = \mathcal{O}(\sqrt{(\log p)/Th})$, the minimum absolute non-zero entry $a_{\min}$ of $\boldsymbol{A}^{t^*}$ be sufficiently large ($a_{\min} \geq 2\lambda$). If $h = \mathcal{O}(T^{1/3})$ and $\frac{s \log p}{Th} = o(1)$ then*

$$\mathbb{P}[\text{supp}(\hat{\boldsymbol{A}}^{t^*}) = \text{supp}(\boldsymbol{A}^{t^*})] \rightarrow 1, \quad T \rightarrow \infty, \quad \forall t^* \in [0, 1]. \tag{6}$$

# 7 Experiments

To the best of our knowledge, this is the first practical method for structure learning of non-stationary DBNs. Thus we mainly compare with static DBN structure learning methods. The goal is to demonstrate the advantage of TV-DBNs for modeling time-varying structures of non-stationary processes which are ignored by traditional approaches. We conducted 3 experiments using synthetic data, gene expression data and EEG signals. In these experiments, TV-DBNs either better recover the underlying networks, or provide better explanatory power for the underlying biological processes.

**Synthetic Data** In this experiment, we generate synthetic time series using a first order autoregressive models with smoothly varying model structures. More specifically, we first generate 8 different anchor transition matrices $A^{t_1} \dots A^{t_8}$, each of which corresponds to an Erdös-Rényi random graph of node size $p = 50$ and average indegree of 2 (we have also experimented with $p = 75$ and $100$ which provides similar results). We then evenly space these 8 anchor matrices, and interpolate a suitable number of intermediate matrices to match the number of observations $T$. Due to the interpolation, the average indegree of each node is around 4. With the sequence of $\{A^t\}(t = 1 \dots T)$, we simulate the time series according to equation (4) with noise variance $\sigma^2 = 1$. We then study the behavior of TV-DBNs and static DBNs [22] in recovering the underlying varying networks as we increase the number of observations $T$. We also compare with a piecewise constant DBN that estimate a static network for each segment obtained from change point detection [14].

For the TV-DBN, we choose the bandwidth parameter $h$ of the Gaussian kernel according to the spacing between two adjacent anchor matrices $(T/7)$ such that $\exp(-\frac{T^2}{49h}) = \exp(-1)$. For all methods, we choose the regularization parameter such that the resulting networks has an average indegree of 4. We evaluate the performance using an F1 score, which is the harmonic mean of precision and recall scores in retrieving the true time-varying network edges.

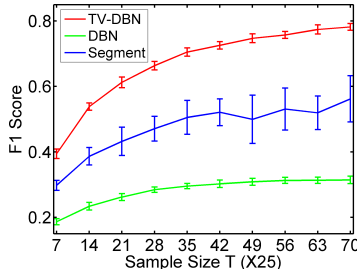

We can see that estimating a static DBN or a piecewise constant DBN does not provide a good estimation of the network structures (Figure 1). In contrast, the TV-DBN leads to a significantly higher F1 score, and its performance also benefit quickly from increasing the number of observations. Note that these results are not surprising since time-varying networs simply fit better with the data generating process. As time-varying networks occur often in biological systems, we expect TV-DBNs will be useful for studying biological systems.

Figure 1: F1 score of estimating time-varying networks for different methods.

**Yeast Gene Regulatory Networks.** In this experiment, we will reverse engineer the time varying gene regulatory networks from time series of gene expression measured across two yeast cell cycles. A yeast cell cycle is divided into four stages: S phase for DNA synthesis, M phase for mitosis, and G1 and G2 phase separating S and M phases. We use two time series (alpha30 and alpha38) from [23] which are technical replicates of each other with a sampling interval of 5 minutes and a total of 25 time points across two yeast cell cycles. We consider a set of 3626 genes which are common to both arrays. We choose the bandwidth parameter $h$ such that the weighting decay to $\exp(-1)$ for half of a cell cycle, *i.e.* $\exp(-6^2/h) = \exp(-1)$. We choose the regularization parameter such that the sparsity of the networks are around $0.01$.

During the cell cycle of yeasts, there exist multiple underlying "themes" that determine the functionalities of each gene and their relationships to each other, and such themes are dynamical and stochastic. As a result, the gene regulatory networks at each time point are context-dependent and can undergo systematic rewiring, rather than being invariant over time. A summary of the estimated time-varying networks are visualized in Figure 2. We group genes according to 50 ontology groups. We can see that the most active groups of genes are related to background processes such as cytoskeleton organization, enzyme regulator activity, ribosome activity. We can also spot transient interactions, for instance, between genes related to site of polarized growth and nucleolus (time point 18), and between genes related to ribosome and cellular homeostasis (time point 24). Note that, although gene expressions are measured across two cell cycles, the values do not necessarily exhibit periodic behavior. In fact, only a small fraction of yeast genes (less than 20%) has been reported to exhibit cycling behavior [23].

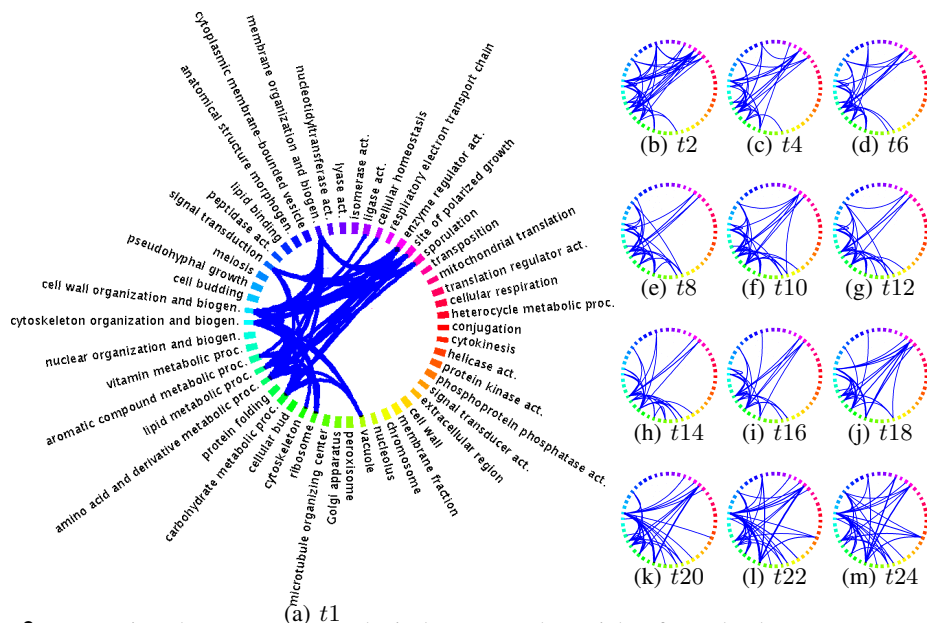

(a) $t1$

Figure 2: Interactions between gene ontological groups. The weight of an edge between two ontological groups is the total number of connection between genes in the two groups. We thresholded the edge weight such that only the dominant interactions are displayed.

Next we study genes sets that are related to specific stage of cell cycle where we expect to see periodic behavior. In particular, we obtain gene sets known to be related to G1, S and S/G2 stage respectively.[1] We use interactivity, which is the total number of edges a group of genes is connected to, to describe the activity of each group of genes. Since the regulatory networks are directed, we can examine both indegree and out-

Table 1: The number of enriched unique gene sets discovered by the static and time-varying networks respectively. Here we are interested in recall score: the time-varying networks better models the biological system.

|  | DBN | TV-DBN |
|---|---|---|
| TF | 7 | 23 |
| Knockout | 7 | 26 |
| Ontology | 13 | 77 |

degree separately for each gene sets. In Figure 3(a)(b)(c), the interactivities of these genes indeed exhibit periodic behavior which corresponds well with their supposed functions in cell cycles.

We also plot the histogram of indegree and outdegree (averaged across time) for the time-varying networks in Figure 3(d). We find that the outdegrees approximately follow a scale free distribution with largest outdegree reaching 90. This corresponds well with the biological observation that there are a few genes (regulators) that regulate a lot of other genes. The indegree distribution is very different from that of the outdegree, and it exhibits a clear peak between 5 and 6. This also corresponds well with biological observations that most genes are controlled only by a few regulators.

To further assess the modeling power of the time-varying networks and its advantage over static network, we perform gene set enrichment studies. More specifically, we use three types of information to define the gene sets: transcription factor binding targets (TF), gene knockout signatures (Knockout), and gene ontology (Ontology) groups [24]. We partition the genes in the time varying networks at each time point into 50 groups using spectral clustering, and then test whether these groups are enriched with genes from any predefined gene sets. We use a max-statistic and a 99% confidence level for the test [25]. Table 1 indicates that time-varying networks are able to discover more functional groups as defined by the genes sets than static networks as commonly used in biological literature. In the appendix, we also visualize the time spans of these active functional groups. It can be seen that many of them are dynamic and transient, and not captured by a static network.

**Brain Response to Visual Stimuli.** In this experiment, we will explore the interactions between brain regions in response to visual stimuli using TV-DBNs. We use the EEG dataset from [26] where five healthy subjects (labeled 'aa', 'al', 'av', 'aw' and 'ay' respectively) were required to imagine body part movement based on visual cues in order to generate EEG changes. We focus our

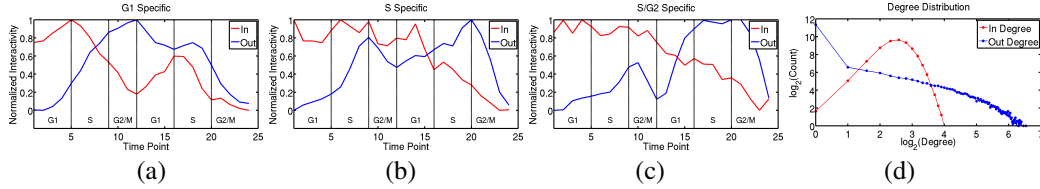

Figure 3: (a) Genes specific to G1 phase are being regulated periodically; we can see that the average in-degree of these genes increases during G1 stage and starts to decline right after the G1 phase. (b) S phase specific genes periodically regulate other genes; we can see that the average outdegree of these genes peaks at the end of S phase and starts to decline right after S phase. (c) The interactivity of S/G2 specific genes also show nice correspondence with their functional roles; we can see that the average outdegree increases till G2 phase and then starts to decline. (d) Indegree and outdegree distribution averaged over 24 time points.

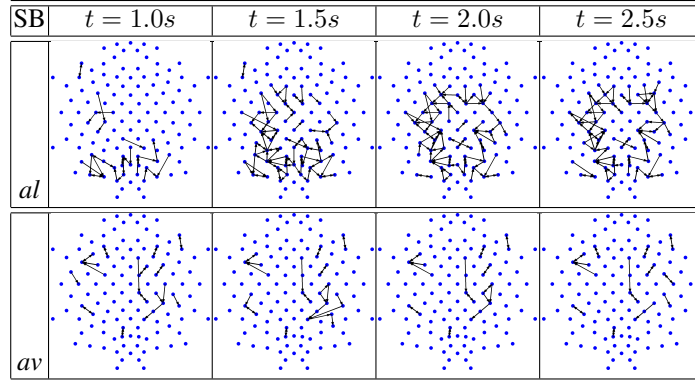

Figure 4: Temporal progression of brain interactions for subject 'al' and BCI "illiterate" 'av'. The plot for the other 3 subjects can be found in the appendix. The dots correspond to EEG electrode positions in 10-5 system.

analysis on trials related to right hand imagination, and signals in the window $[1.0, 2.5]$ second after the visual cue is presented. We bandpass filter data at 8–12 Hz to obtain EEG alpha activity. We further normalize each EEG channel to zero mean and unit variance, and estimate the time-varying networks for all 5 subject using exactly the same regularization parameter and kernel bandwidth ($h$ s.t. $\exp(-(0.5)^2/h) = \exp(-1)$). We tried a range of different regularization parameters, but obtained qualitatively similar results to Figure 4.

What is particularly interesting in this dataset is that subject 'av' is called BCI "illiterate"; he/she is unable to generate clear EEG changes during motor imagination. The estimated time-varying networks reveal that the brain interactions of subject 'av' is particularly weak and the brain connectivity actually decreases as the experiment proceeds. In contrast, all other four subjects show an increased brain interaction as they engage in active imagination. Particularly, these increased interactions occur between visual and motor cortex. This dynamic coherence between visual and motor cortex corresponds nicely to the fact that subjects are consciously transforming visual stimuli into motor imaginations which involves the motor cortex. It seems that subject 'av' fails to perform such integration due to the disruption of brain interactions.

## 8 Conclusion

In this paper, we propose time-varying dynamic Bayesian networks (TV-DBN) for modeling the varying network structures underlying non-stationary biological time series. We have designed a simple and scalable kernel reweighted structural learning algorithm to make the learning possible. Given the rapid advances in data collection technologies for biological systems, we expect that complex, high-dimensional, and feature rich data from complex dynamic biological processes, such as cancer progression, immune responses, and developmental processes, will continue to grow. Thus, we believe our new method is a timely contribution that can narrow the gap between imminent methodological needs and the available data and offer deeper understanding of the mechanisms and processes underlying biological networks.

**Acknowledgments** LS is supported by a Ray and Stephenie Lane Research Fellowship. EPX is supported by grant ONR N000140910758, NSF DBI-0640543, NSF DBI-0546594, NSF IIS-0713379 and an Alfred P. Sloan Research Fellowship. We also thank Grace Tzu-Wei Huang for helpful discussions.

## Footnotes

[1]We obtain gene sets from http://genome-www.stanford.edu/cellcycle/data/rawdata/KnowGenes.doc.

# References

[1] A. L. Barabasi and Z. N. Oltvai. Network biology: Understanding the cell's functional organization. *Nature Reviews Genetics*, 5(2):101–113, 2004.

[2] Francisco Varela, Jean-Philippe Lachaux, Eugenio Rodriguez, and Jacques Martinerie. The brainweb: Phase synchronization and large-scale integration. *Nature Reviews Neuroscience*, 2:229–239, 2001.

[3] N. Luscombe, M. Babu, H. Yu, M. Snyder, S. Teichmann, and M. Gerstein. Genomic analysis of regulatory network dynamics reveals large topological changes. *Nature*, 431:308–312, 2004.

[4] Eugenio Rodriguez, Nathalie George, Jean-Philippe Lachaux, Jacques Martinerie, Bernard Renault, and Francisco J. Varela1. Perception's shadow: long-distance synchronization of human brain activity. *Nature*, 397(6718):430–433, 1999.

[5] M. Talih and N. Hengartner. Structural learning with time-varying components: Tracking the cross-section of financial time series. *J. Royal Stat. Soc. B*, 67(3):321C341, 2005.

[6] S. Hanneke and E. P. Xing. Discrete temporal models of social networks. In *Workshop on Statistical Network Analysis, ICML06*, 2006.

[7] F. Guo, S. Hanneke, W. Fu, and E. P. Xing. Recovering temporally rewiring networks: A model-based approach. In *International Conference in Machine Learning*, 2007.

[8] X. Xuan and K. Murphy. Modeling changing dependency structure in multivariate time series. In *International Conference in Machine Learning*, 2007.

[9] J. Robinson and A. Hartemink. Non-stationary dynamic bayesian networks. In *Neural Information Processing Systems*, 2008.

[10] Amr Ahmed and Eric P. Xing. Tesla: Recovering time-varying networks of dependencies in social and biological studies. *Proceeding of the National Academy of Sciences*, in press, 2009.

[11] S. Zhou, J. Lafferty, and L. Wasserman. Time varying undirected graphs. In *Computational Learning Theory*, 2008.

[12] L. Song, M. Kolar, and E. Xing. Keller: Estimating time-evolving interactions between genes. In *Bioinformatics (ISMB)*, 2009.

[13] N. Friedman, M. Linial, I. Nachman, and D. Peter. Using bayesian networks to analyze expression data. *Journal of Computational Biology*, 7:601–620, 2000.

[14] N. Dobingeon, J. Tourneret, and M. Davy. Joint segmentation of piecewise constant autoregressive processes by using a hierarchical model and a bayesian sampling approach. *IEEE Transactions on Signal Processing*, 55(4):1251–1263, 2007.

[15] K. Kanazawa, D. Koller, and S. Russell. Stochastic simulation algorithms for dynamic probabilistic networks. *Uncertainty in AI*, 1995.

[16] L. Getoor, N. Friedman, D. Koller, and B. Taskar. Learning probabilistic models with link uncertainty. *Journal of Machine Learning Research*, 2002.

[17] R. Dahlhaus. Fitting time series models to nonstationary processes. *Ann. Statist*, (25):1–37, 1997.

[18] C. Andrieu, M. Davy, and A. Doucet. Efficient particle filtering for jump markov systems: Application to time-varying autoregressions. *IEEE Transactions on Signal Processing*, 51(7):1762–1770, 2003.

[19] E. H. Davidson. *Genomic Regulatory Systems*. Academic Press, 2001.

[20] Florentina Bunea. Honest variable selection in linear and logistic regression models via $\ell_1$ and $\ell_1 + \ell_2$ penalization. *Electronic Journal of Statistics*, 2:1153, 2008.

[21] W. Fu. Penalized regressions: the bridge versus the lasso. *Journal of Computational and Graphical Statistics*, 7(3):397–416, 1998.

[22] M. Schmidt, A. Niculescu-Mizil, and K Murphy. Learning graphical model structure using l1-regularization paths. In *AAAI*, 2007.

[23] Tata Pramila, Wei Wu, Shawna Miles, William Noble, and Linda Breeden. The forkhead transcription factor hcm1 regulates chromosome segregation genes and fills the s-phase gap in the transcriptional circuitry of the cell cycle. *Gene and Development*, 20:2266–2278, 2006.

[24] Jun Zhu, Bin Zhang, Erin Smith, Becky Drees, Rachel Brem, Leonid Kruglyak, Roger Bumgarner, and Eric E Schadt. Integrating large-scale functional genomic data to dissect the complexity of yeast regulatory networks. *Nature Genetics*, 40:854–861, 2008.

[25] T. Nichols and A. Holmes. Nonparametric permutation tests for functional neuroimaging: a primer with examples. *Human Brain Mapping*, 15:1–25, 2001.

[26] G. Dornhege, B. Blankertz, G. Curio, and K.R. Müller. Boosting bit rates in non-invasive eeg single-trial classifications by feature combination and multi-class paradigms. *IEEE Trans. Biomed. Eng.*, 51:993–1002, 2004.

